# Learning a Gaussian Process Prior for Automatically Generating Music Playlists

**John C. Platt**      **Christopher J. C. Burges**
**Steven Swenson**      **Christopher Weare**      **Alice Zheng** *

Microsoft Corporation
1 Microsoft Way
Redmond, WA 98052
{*jplatt,cburges,sswenson,chriswea*} *@microsoft.com, alicez@cs.berkeley.edu*

## Abstract

This paper presents AutoDJ: a system for automatically generating music playlists based on one or more seed songs selected by a user. AutoDJ uses Gaussian Process Regression to learn a user preference function over songs. This function takes music metadata as inputs. This paper further introduces Kernel Meta-Training, which is a method of learning a Gaussian Process kernel from a distribution of functions that generates the learned function. For playlist generation, AutoDJ learns a kernel from a large set of albums. This learned kernel is shown to be more effective at predicting users' playlists than a reasonable hand-designed kernel.

## 1   Introduction

Digital music is becoming very widespread, as personal collections of music grow to thousands of songs. One typical way for a user to interact with a personal music collection is to specify a *playlist*, an ordered list of music to be played. Using existing digital music software, a user can manually construct a playlist by individually choosing each song. Alternatively, playlists can be generated by the user specifying a set of rules about songs (e.g., genre = rock), and the system randomly choosing songs that match those rules.

Constructing a playlist is a tedious process: it takes time to generate a playlist that matches a particular mood. It is also difficult to construct a playlist in advance, as a user may not anticipate all possible music moods and preferences he or she will have in the future.

AutoDJ is a system for automatically generating playlists at the time that a user wants to listen to music. The playlist plays with minimal user intervention: the user hears music that is suitable for his or her current mood, preferences and situation.

AutoDJ has a simple and intuitive user interface. The user selects one or more seed songs for AutoDJ to play. AutoDJ then generates a playlist with songs that are similar to the seed songs. The user may also review the playlist and add or remove certain songs, if they don't fit. Based on this modification, AutoDJ then generates a new playlist.

AutoDJ uses a machine learning system that finds a current user preference function $f$ over a feature space of music. Every time a user selects a seed song or removes a song from the

playlist, a training example is generated. In general, a user can give an arbitrary preference value to any song. By default, we assume that selected songs have target $f$ values of 1, while removed songs have target $f$ values of 0. Given a training set, a full user preference function $f$ is inferred by regression. The $f$ for each song owned by the user is evaluated, and the songs with the highest $f$ are placed into the playlist.

The machine learning problem defined above is difficult to solve well. The training set often contains only one training example: a single seed song that the user wishes to listen to. Most often, AutoDJ must infer an entire function from 1–3 training points. An appropriate machine learning method for such small training sets is Gaussian Process Regression (GPR) [14], which has been shown empirically to work well on small data sets. Technical details of how to apply GPR to playlist generation are given in section 2. In broad detail, GPR starts with a similarity or kernel function $K(\mathbf{x}, \mathbf{x}')$ between any two songs. We define the input space $\mathbf{x}$ to be descriptive metadata about the song. Given a training set of user preferences, a user preference function is generated by forming a linear blend of these kernel functions, whose weights are solved via a linear system. This user preference function is then used to evaluate all of the songs in the user's collection.

This paper introduces a new method of generating a kernel for use in GPR. We call this method Kernel Meta-Training (KMT). Technical details of KMT are described in section 3. KMT improves GPR by adding an additional phase of learning: meta-training. During meta-training, a kernel is learned before any training examples are available. The kernel is learned from a set of samples from meta-training functions. These meta-training functions are drawn from the same function distribution that will eventually generate the training function. In order to generalize the kernel beyond the meta-training data set, we fit a parameterized kernel to the meta-training data, with many fewer parameters than data points. The kernel is parameterized as a non-negative combination of base Mercer kernels. These kernel parameters are tuned to fit the samples across the meta-training functions. This constrained fit leads to a simple quadratic program. After meta-training, the kernel is ready to use in standard GPR.

To use KMT to generate playlists, we meta-train a kernel on a large number of albums. The learned kernel thus reflects the similarity of songs on professionally designed albums. The learned kernel is hardwired into AutoDJ. GPR is then performed using the learned kernel every time a user selects or removes songs from a playlist. The learned kernel forms a good prior, which enables AutoDJ to learn a user preference function with a very small number of user training examples.

## 1.1 Previous Work

There are several commercial Web sites for playing or recommending music based on one seed song. The algorithms behind these sites are still unpublished.

This work is related to Collaborative Filtering (CF) [9] and to building user profiles in textual information retrieval [11]. However, CF does not use metadata associated with a media object, hence CF will not generalize to new music that has few or no user votes. Also, no work has been published on building user profiles for music. The ideas in this work may also be applicable to text retrieval.

Previous work in GPR [14] learned kernel parameters through Bayesian methods from just the training set, not from meta-training data. When AutoDJ generates playlists, the user may select only one training example. No useful similarity metric can be derived from one training example, so AutoDJ uses meta-training to learn the kernel.

The idea of meta-training comes from the "learning to learn" or multi-task learning literature [2, 5, 10, 13]. This paper is most similar to Minka & Picard [10], who also suggested fitting a mean and covariance for a Gaussian Process based on related functions. However, in [10], in order to generalize the covariance beyond the meta-training points, a Multi-Layer Perceptron (MLP) is used to learn multiple tasks, which requires non-convex optimization.

The Gaussian Process is then extracted from the MLP. In this work, using a quadratic program, we fit a parameterized Mercer kernel directly to a meta-training kernel matrix in order to generalize the covariance.

Meta-training is also related to algorithms that learn from both labeled and unlabeled data [3, 6]. However, meta-training has access to more data than simply unlabeled data: it has access to the values of the meta-training functions. Therefore, meta-training may perform better than these other algorithms.

## 2 Gaussian Process Regression for Playlist Generation

AutoDJ uses GPR to generate a playlist every time a user selects one or more songs. GPR uses a Gaussian Process (GP) as a prior over functions. A GP is a stochastic process $Y(\mathbf{x})$ over a multi-dimensional input space $\mathbf{x}$. For any $N$, if $N$ vectors $\mathbf{x}_i$ are chosen in the input space, and the $N$ corresponding samples $y_i$ are drawn from the GP, then the $y_i$ are jointly Gaussian.

There are two statistics that fully describe a GP: the mean $\mu(\mathbf{x})$ and the covariance $K(\mathbf{x}, \mathbf{x}')$. In this paper, we assume that the GP over user preference functions is zero mean. That is, at any particular time, the user does not want to listen to most of the songs in the world, which leads to a mean preference close enough to zero to approximate as zero. Therefore, the covariance kernel $K(\mathbf{x}, \mathbf{x}')$ simply turns into a correlation over a distribution of functions $g$: $K(\mathbf{x}, \mathbf{x}') = < g(\mathbf{x})g(\mathbf{x}') >$.

In section 3, we learn a kernel $K(\mathbf{x}, \mathbf{x}')$ which takes music metadata as $\mathbf{x}$ and $\mathbf{x}'$. In this paper, whenever we refer to a music metadata vector, we mean a vector consisting of 7 categorical variables: genre, subgenre, style, mood, rhythm type, rhythm description, and vocal code. This music metadata vector is assigned by editors to every track of a large corpus of music CDs. Sample values of these variables are shown in Table 1. Our kernel function $K(\mathbf{x}, \mathbf{x}')$ thus computes the similarity between two metadata vectors corresponding to two songs. The kernel only depends on whether the same slot in the two vectors are the same or different. Specific details about the kernel function are described in section 3.2.

| Metadata Field | Example Values | Number of Values |
|---|---|---|
| Genre | Jazz, Reggae, Hip-Hop | 30 |
| Subgenre | Heavy Metal, I'm So Sad and Spaced Out | 572 |
| Style | East Coast Rap, Gangsta Rap, West Coast Rap | 890 |
| Mood | Dreamy, Fun, Angry | 21 |
| Rhythm Type | Straight, Swing, Disco | 10 |
| Rhythm Description | Frenetic, Funky, Lazy | 13 |
| Vocal Code | Instrumental, Male, Female, Duet | 6 |

Table 1: Music metadata fields, with some example values

Once we have defined a kernel, it is simple to perform GPR. Let $\mathbf{x}_i$ be the metadata vectors for the $N$ songs for which the user has expressed a preference by selecting or removing them from the playlist. Let $t_i$ be the expressed user preference. In general, $t_i$ can be any real value. If the user does not express a real-valued preference, $t_i$ is assumed 1 if the user wants to listen to the song and 0 if the user does not. Even if the values $t_i$ are binary, we do not use Gaussian Process Classification (GPC), in order to maintain generality and because GPC requires an iterative procedure to estimate the posterior [1].

Let $f_i$ be the underlying true user preference for the $i$th song, of which $t_i$ is a noisy measurement, with Gaussian noise of variance $\sigma^2$. Also, let $\mathbf{x}_*$ be a metadata vector of any song that will be considered to be on a playlist: $f_*$ is the (unknown) user preference for that song.

Before seeing the preferences $t_i$, the vector $[f_i, f_*]$ forms a joint prior Gaussian derived from the GP. After incorporating the $t_i$ information, the posterior mean of $f_*$ is

$$f_* = \sum_{i=1}^{N} \alpha_i K(\mathbf{x}_i, \mathbf{x}_*), \tag{1}$$

where $\alpha_i = \sum_j A_{ij} t_j$ and

$$A_{ij} = \left( K(\mathbf{x}_i, \mathbf{x}_j) + \sigma^2 \delta_{ij} \right)^{-1}. \tag{2}$$

Thus, the user preference function for a song s, $f(s)$, is a linear blend of kernels $K(\mathbf{x}_i, \mathbf{x}(s))$ that compare the metadata vector for song $s$ with the metadata vectors $\mathbf{x}_i$ for the songs that the user expressed a preference. The weights $\alpha_i$ are computed by inverting an $N$ by $N$ matrix. Since the number of user preferences $N$ tends to be small, inverting this matrix is very fast.

Since the kernel is learned before GPR, and the vector $t_i$ is supplied by the user, the only free hyperparameter is the noise value $\sigma$. This hyperparameter is selected via maximum likelihood on the training set. The formula for the log likelihood of the training data given $\sigma$ is

$$\log p(\text{data}|\sigma) = 0.5 \log |A| - 0.5 \mathbf{t}^T A \mathbf{t} - 0.5 N \log 2\pi. \tag{3}$$

Every time a playlist is generated, different values of $\sigma$ are evaluated and the $\sigma$ that generates the highest log likelihood is used.

In order to generate the playlist, the matrix $A$ is computed, and the user preference function $f(s)$ is computed for every song that the user owns. The songs are then ranked in descending order of $f$. The playlist consists of the top songs in the ranked list. The playlist can cut off after a fixed number of songs, e.g., 30. It can also cut off if the value of $f$ gets too low, so that the playlist only contains songs that the user will enjoy.

The order of the playlist is the order of the songs in the ranked list. This is empirically effective: the playlist typically starts with the selected seed songs, proceeds to songs very similar to the seed songs, and then gradually drifts away from the seed songs towards the end of the list, when the user is paying less attention. We explored neural networks and SVMs for determining the order of the playlist, but have not found a clearly more effective ordering algorithm than simply the order of $f$. Here, "effective" is defined as generating playlists that are pleasing to the authors.

## 3    Kernel Meta-Training (KMT)

This section describes Kernel Meta-Training (KMT) that creates the GP kernel $K(\mathbf{x}, \mathbf{x}')$ used in the previous section. As described in the introduction, KMT operates on samples drawn from a set of $M$ functions $f_m(\mathbf{x})$. This set of functions should be related to a final trained function, since we derive a similarity kernel from the meta-training set of functions. In other words, we learn a Gaussian prior over the space of functions by computing Gaussian statistics on a set of functions related to a function we wish to learn.

We express the kernel $K$ as a covariance components model [12]:

$$K(\mathbf{x}, \mathbf{x}') = \sum_{n=1}^{N_\psi} \beta_n \psi_n(\mathbf{x}, \mathbf{x}'), \tag{4}$$

where $\psi_n$ are pre-defined Mercer kernels and $\beta_n \geq 0$. We then fit $\beta_n$ to the samples drawn from the meta-training functions. We use the simpler model instead of an empirical covariance matrix, in order to generalize the GPR beyond points that are in the meta-training set.

The functional form of the kernel $\psi$ and $N_\psi$ can be chosen via cross-validation. In our application, both the form of $\psi$ and $N_\psi$ are determined by the available input data (see section 3.2, below).

One possible method to fit the $\beta_n$ is to maximize the likelihood in (3) over all samples drawn from all meta-training functions [7]. However, solving for the optimal $\beta_n$ requires an iterative algorithm whose inner loop requires Cholesky decomposition of a matrix of dimension the number of meta-training samples. For our application, this matrix would have dimension 174,577, which makes maximizing the likelihood impractical.

Instead of maximizing the likelihood, we fit a covariance components model to an empirical covariance computed on the meta-training data set, using a least-square distance function:

$$\arg\min_{\beta_n} \frac{1}{2} \sum_{i,j} \left( K_{ij} - \sum_{n=1}^{N_\psi} \beta_n \psi_n(\mathbf{x}_i, \mathbf{x}_j) \right)^2, \tag{5}$$

where $i$ and $j$ index all of the samples in the meta-training data set, and where $K_{ij}$ is the empirical covariance

$$K_{ij} = \frac{1}{M} \sum_{m=1}^{M} f_m(\mathbf{x}_i) f_m(\mathbf{x}_j). \tag{6}$$

In order to ensure that the final kernel in (4) is Mercer, we apply $\beta_n \geq 0$ as a constraint in optimization. Solving (5) subject to non-negativity constraints results in a fast quadratic program of size $N_\psi$. Such a quadratic program can be solved quickly and robustly by standard optimization packages.

The cost function in equation (5) is the square of the Frobenius norm of the difference between the empirical matrix $K_{ij}$ and the fit kernel $K(\beta)$. The use of the Frobenius norm is similar to the Ordinary Least Squares technique of fitting variogram parameters in geostatistics [7]. However, instead of summing variogram estimates within spatial bins, we form covariance estimates over all meta-training data pairs $i, j$.

Analogous to [8], we can prove that the Frobenius norm is consistent: as the amount of training data goes to infinity, the empirical Frobenius norm, above, approaches the Frobenius norm of the difference between the true kernel and our fit kernel. (The proof is omitted to save space). Finally, unlike the cost function presented in [8], the cost function in equation (5) produces an easy-to-solve quadratic program.

### 3.1  KMT for Music Playlist Generation

In this section, we consider the application of the general KMT technique to music playlist generation.

We decided to use albums to generate a prior for playlist generation, since albums can be considered to be professionally designed playlists. For the meta-training function $g_k$, we use album indicator functions that are 1 for songs on an album $k$, and 0 otherwise. Thus, KMT learns a similarity metric that professionals use when they assemble albums. This same similarity metric empirically makes consonant playlists. Using a small $N_\psi$ in equation (4) forces a smoother, more general similarity metric. If we had simply used the meta-training kernel matrix $K_{ij}$ without fitting $K(\beta)$, the playlist generator would exactly reproduce one or more albums in the meta-training database. This is the meta-training equivalent of overfitting.

Because the album indicator functions are uniquely defined for songs, not for metadata vectors, we cannot simply generate a kernel matrix according to (6). Instead, we generate a meta-training kernel matrix using meta-training functions that depend on songs:

$$K_{ij} = \frac{1}{M} \sum_{k=1}^{M} g_k(i) g_k(j), \tag{7}$$

where $g_k(i)$ is 1 if song $i$ belongs to album $k$, 0 otherwise. We then fit the $\beta_n$ according to (5), where the $\psi_n$ Mercer kernels depend on music metadata vectors $\mathbf{x}$ that are defined in

Table 1. The resulting kernel is still defined by (4), with a specific $\psi_n$ that will be defined in section 3.2, below.

We used 174,577 songs and 14,198 albums to make up the meta-training matrix $K_{ij}$, which is dimension 174,577x174,577. However, note that the $K_{ij}$ meta-training matrix is very sparse, since most songs only belong to 1 or 2 albums. Therefore, it can be stored as a sparse matrix. We use a quadratic programming package in Matlab that requires the constant and linear parts of the gradient of the cost function in (5):

$$\frac{\partial E}{\partial \beta_m} = \sum_{i,j} \left( K_{ij} - \sum_n \beta_n \psi_n(\mathbf{x}_i, \mathbf{x}_j) \right) \psi_m(\mathbf{x}_i, \mathbf{x}_j) \tag{8}$$

$$= \sum_{(i,j) \in A} \psi_m(\mathbf{x}_i, \mathbf{x}_j) + \sum_n \beta_n \sum_{i,j} \psi_n(\mathbf{x}_i, \mathbf{x}_j) \psi_m(\mathbf{x}_i, \mathbf{x}_j), \tag{9}$$

where the first (constant) term is only evaluated on those indicies $(i, j)$ in the set $A$ of non-zero $K_{ij}$. The second (linear) term requires a sum over all $i$ and $j$, which is impractical. Instead, we estimate the second term by sampling a random subset of $(i, j)$ pairs (100 random $j$ for each $i$).

### 3.2 Kernels for Categorical Data

The kernel learned in section 3 must operate on categorical music metadata. Up until now, kernels have been defined to operate on continuous data. We could convert the categorical data to a vector space by allocating one dimension for every possible value of each categorical variable, using a 1-of-N sparse code. This would lead to a vector space of dimension 1542 (see Table 1) and would produce a large number of kernel parameters. Hence, we have designed a new kernel that operates directly on categorical data.

We define a family of Mercer kernels:

$$\psi_n(\mathbf{x}, \mathbf{x}') = \begin{cases} 1, & \text{if } a_{nl} = 0 \text{ or } x_l = x_l' \quad \forall l; \\ 0, & \text{otherwise,} \end{cases} \tag{10}$$

where $\mathbf{a}_n$ is defined to be the binary representation of the number $n$. The $\mathbf{a}_n$ vector serves as a mask: when $a_{nl}$ is 1, then the $l$th component of the two vectors must match in order for the output of the kernel to be 1. Due to space limitations, proof of the Mercer property of this kernel is omitted.

For playlist generation, the $\psi_n$ operate on music metadata vectors $\mathbf{x}$ that are defined in Table 1. These vectors have 7 fields, thus $l$ runs from 1 to 7 and $n$ runs from 1 to 128. Therefore, there are 128 free parameters in the kernel which are fit according to (5). The sum of 128 terms in (4) can be expressed as a single look-up table, whose keys are 7-bit long binary vectors, the $l$th bit corresponding to whether $x_l = x_l'$. Thus, the evaluation of $f$ from equation (1) on thousands of pieces of music can be done in less than a second on a modern PC.

## 4 Experimental Results

We have tested the combination of GPR and KMT for the generation of playlists. We tested AutoDJ on 60 playlists manually designed by users in Microsoft Research. We compared the full GPR + KMT AutoDJ with simply using GPR with a pre-defined kernel, and without using GPR and with a pre-defined kernel (using (1) with all $\alpha_i$ equal). We also compare to a playlist which are all of the user's songs permuted in a random order. As a baseline, we decided to use Hamming distance as the pre-defined kernel. That is, the similarity between two songs is the number of metadata fields that they have in common.

We performed tests using only positive training examples, which emulates users choosing seed songs. There were 9 experiments, each with a different number of seed songs, from 1 to 9. Let the number of seed songs for an experiment be $S$. Each experiment consisted

of 1000 trials. Each trial chose a playlist at random (out of the playlists that consisted of at least $S + 1$ songs), then chose $S$ songs at random out of the playlist as a training set. The test set of each trial consisted of all of the remaining songs in the playlist, plus all other songs owned by the designer of the playlist. This test set thus emulates the possible songs available to the playlist generator.

To score the produced playlists, we use a standard collaborative filtering metric, described in [4]. The score of a playlist for trial $j$ is defined to be

$$R_j = \sum_{i=1}^{N_j} \frac{t_{ij}}{2^{(i-1)/(\beta-1)}}, \tag{11}$$

where $t_{ij}$ is the user preference of the $i$th element of the $j$th playlist (1 if $i$th element is on playlist $j$, 0 otherwise), $\beta$ is a "half-life" of user interest in the playlist (set here to be 10), and $N_j$ are the number of test songs for playlist $j$. This score is summed over all 1000 trials, and normalized:

$$R = 100 \sum_{j=1}^{1000} R_j \Big/ \sum_{j=1}^{1000} R_j^{\mathrm{max}}, \tag{12}$$

where $R_j^{\mathrm{max}}$ is the score from (11) if that playlist were perfect (i.e., all of the true playlist songs were at the head of the list). Thus, an $R$ score of 100 indicates perfect prediction.

|  | Number of Seed Songs | | | | | | | | |
|---|---|---|---|---|---|---|---|---|---|
| Playlist Method | 1 | 2 | 3 | 4 | 5 | 6 | 7 | 8 | 9 |
| KMT + GPR | **42.9** | **46.0** | **44.8** | 43.8 | **46.8** | **45.0** | **44.2** | **44.4** | **44.8** |
| Hamming + GPR | 32.7 | 39.2 | 39.8 | 39.6 | 41.3 | 40.0 | 39.5 | 38.4 | 39.8 |
| Hamming + No GPR | 32.7 | 39.0 | 39.6 | 40.2 | 42.6 | 41.4 | 41.5 | 41.7 | 43.2 |
| Random Order | 6.3 | 6.6 | 6.5 | 6.2 | 6.5 | 6.6 | 6.2 | 6.1 | 6.8 |

Table 2: $R$ Scores for Different Playlist Methods. Boldface indicates best method with statistical significance level $p < 0.05$.

The results for the 9 different experiments are shown in Table 2. A boldface result shows the best method based on pairwise Wilcoxon signed rank test with a significance level of 0.05 (and a Bonferroni correction for 6 tests).

There are several notable results in Table 2. First, all of the experimental systems perform much better than random, so they all capture some notion of playlist generation. This is probably due to the work that went into designing the metadata schema. Second, and most importantly, the kernel that came out of KMT is substantially better than the hand-designed kernel, especially when the number of positive examples is 1–3. This matches the hypothesis that KMT creates a good prior based on previous experience. This good prior helps when the training set is extremely small in size. Third, the performance of KMT + GPR saturates very quickly with number of seed songs. This saturation is caused by the fact that exact playlists are hard to predict: there are many appropriate songs that would be valid in a test playlist, even if the user did not choose those songs. Thus, the quantitative results shown in Table 2 are actually quite conservative.

|  | Playlist 1 | Playlist 2 |
|---|---|---|
| Seed | Eagles, The Sad Cafe | Eagles, Life in the Fast Lane |
| 1 | Genesis, More Fool Me | Eagles, Victim of Love |
| 2 | Bee Gees, Rest Your Love On Me | Rolling Stones, Ruby Tuesday |
| 3 | Chicago, If You Leave Me Now | Led Zeppelin, Communication Breakdown |
| 4 | Eagles, After The Thrill Is Gone | Creedence Clearwater, Sweet Hitch-hiker |
| 5 | Cat Stevens, Wild World | Beatles, Revolution |

Table 3: Sample Playlists

To qualitatively test the playlist generator, we distributed a prototype version of it to a few individuals in Microsoft Research. The feedback from use of the prototype has been very positive. Qualitative results of the playlist generator are shown in Table 3. In that table, two different Eagles songs are selected as single seed songs, and the top 5 playlist songs are shown. The seed song is always first in the playlist and is not repeated. The seed song on the left is softer and leads to a softer playlist, while the seed song on the right is harder rock and leads to a more hard rock play list.

## 5 Conclusions

We have presented an algorithm, Kernel Meta-Training, which derives a kernel from a set of meta-training functions that are related to the function that is being learned. KMT permits the learning of functions from very few training points. We have applied KMT to create AutoDJ, which is a system for automatically generating music playlists. However, the KMT idea may be applicable to other tasks.

Experiments with music playlist generation show that KMT leads to better results than a hand-built kernel when the number of training examples is small. The generated playlists are qualitatively very consonant and useful to play as background music.

## Footnotes

*Current address: Department of Electrical Engineering and Computer Science, University of California at Berkeley

## References

[1] D. Barber and C. K. I. Williams. Gaussian processes for Bayesian classification via hybrid Monte Carlo. In M. C. Mozer, M. I. Jordan, and T. Petsche, editors, *NIPS*, volume 9, pages 340–346, 1997.

[2] J. Baxter. A Bayesian/information theoretic model of bias learning. *Machine Learning*, 28:7–40, 1997.

[3] K. P. Bennett and A. Demiriz. Semi-supervised support vector machines. In M. S. Kearns, S. A. Solla, and D. A. Cohn, editors, *NIPS*, volume 11, pages 368–374, 1998.

[4] J. S. Breese, D. Heckerman, and C. Kadie. Empirical analysis of predictive algorithms for collaborative filtering. In *Uncertainty in Artificial Intelligence*, pages 43–52, 1998.

[5] R. Caruana. Learning many related tasks at the same time with backpropagation. In *NIPS*, volume 7, pages 657–664, 1995.

[6] V. Castelli and T. M. Cover. The relative value of labeled and unlabled samples in pattern recognition with an unknown mixing parameter. *IEEE Trans. Info. Theory*, 42(6):75–85, 1996.

[7] N. A. C. Cressie. *Statistics for Spatial Data*. Wiley, New York, 1993.

[8] N. Cristianini, A. Elisseeff, and J. Shawe-Taylor. On optimizing kernel alignment. Technical Report NC-TR-01-087, NeuroCOLT, 2001.

[9] D. Goldberg, D. Nichols, B. M. Oki, and D. Terry. Using collaborative filtering to weave an information tapestry. *CACM*, 35(12):61–70, 1992.

[10] T. Minka and R. Picard. Learning how to learn is learning with points sets. http://wwwwhite.media.mit.edu/~tpminka/papers/learning.html, 1997.

[11] M. Pazzani and D. Billsus. Learning and revising user profiles: The identification of interesting web sites. *Machine Learning*, 27:313–331, 1997.

[12] P. S. R. S. Rao. *Variance Components Estimation: Mixed models, methodologies and applications*. Chapman & Hill, 1997.

[13] S. Thrun. Is learning the n-th thing any easier than learning the first? In *NIPS*, volume 8, pages 640–646, 1996.

[14] C. K. I. Williams and C. E. Rasmussen. Gaussian processes for regression. In *NIPS*, volume 8, pages 514–520, 1996.
